# BLIND SOURCE SEPARATION VIA MULTINODE SPARSE REPRESENTATION

**Michael Zibulevsky**
Department of Electrical Engineering
Technion, Haifa 32000, Israel
*mzib@ee.technion.ac.il*

**Pavel Kisilev**
Department of Electrical Engineering
Technion, Haifa 32000, Israel
*paulk@tx.technion.ac.il*

**Yehoshua Y. Zeevi**
Department of Electrical Engineering
Technion, Haifa 32000, Israel
*zeevi@ee.technion.ac.il*

**Barak Pearlmutter**
Department of Computer Science
University of New Mexico
Albuquerque, NM 87131 USA
*bap@cs.unm.edu*

## Abstract

We consider a problem of blind source separation from a set of instantaneous linear mixtures, where the mixing matrix is unknown. It was discovered recently, that exploiting the sparsity of sources in an appropriate representation according to some signal dictionary, dramatically improves the quality of separation. In this work we use the property of multiscale transforms, such as wavelet or wavelet packets, to decompose signals into sets of local features with various degrees of sparsity. We use this intrinsic property for selecting the best (most sparse) subsets of features for further separation. The performance of the algorithm is verified on noise-free and noisy data. Experiments with simulated signals, musical sounds and images demonstrate significant improvement of separation quality over previously reported results.

## 1 Introduction

In the blind source separation problem an $N$-channel sensor signal $\mathbf{x}(\xi)$ is generated by $M$ unknown scalar source signals $s_m(\xi)$, linearly mixed together by an unknown $N \times M$ mixing, or crosstalk, matrix $\mathbf{A}$, and possibly corrupted by additive noise $\mathbf{n}(\xi)$:

$$\mathbf{x}(\xi) = \mathbf{A}\mathbf{s}(\xi) + \mathbf{n}(\xi). \qquad (1)$$

The independent variable $\xi$ is either time or spatial coordinates in the case of images. We wish to estimate the mixing matrix $\mathbf{A}$ and the $M$-dimensional source signal $\mathbf{s}(\xi)$.

The assumption of statistical independence of the source components $s_m(\xi)$, $m = 1, ..., M$ leads to the Independent Component Analysis (ICA) [1], [2]. A stronger assumption is the

sparsity of decomposition coefficients, when the sources are properly represented [3]. In particular, let each $s_m(\xi)$ have a sparse representation obtained by means of its decomposition coefficients $c_{mk}$ according to a signal dictionary of functions $\varphi_k(\xi)$:

$$s_m(\xi) = \sum_k c_{mk}\, \varphi_k(\xi). \tag{2}$$

The functions $\varphi_k(\xi)$ are called *atoms* or *elements* of the dictionary. These elements do not have to be linearly independent, and instead may form an overcomplete dictionary, e.g. wavelet-related dictionaries (wavelet packets, stationary wavelets, *etc.*, see for example [9]). Sparsity means that only a small number of coefficients $c_{mk}$ differ significantly from zero. Then, unmixing of the sources is performed in the transform domain, i.e. in the domain of these coefficients $c_{mk}$. The property of sparsity often yields much better source separation than standard ICA, and can work well even with more sources than mixtures. In many cases there are distinct groups of coefficients, wherein sources have different sparsity properties. The key idea in this study is to select only a subset of features (coefficients) which is best suited for separation, with respect to the following criteria: (1) sparsity of coefficients (2) separability of sources' features. After this subset is formed, one uses it in the separation process, which can be accomplished by standard ICA algorithms or by clustering. The performance of our approach is verified on noise-free and noisy data. Our experiments with 1D signals and images demonstrate that the proposed method further improves separation quality, as compared with result obtained by using sparsity of all decomposition coefficients.

## 2 Two approaches to sparse source separation: InfoMax and Clustering

Sparse sources can be separated by each one of several techniques, e.g. the Bell-Sejnowski Information Maximization (BS InfoMax) approach [1], or by approaches based on geometric considerations (see for example [8]). In the former case, the algorithm estimates the *unmixing* matrix $\mathbf{W} = \mathbf{A}^{-1}$, while in the later case the output is the estimated mixing matrix. In both cases, these matrices can be estimated only up to a column permutation and a scaling factor [4].

**InfoMax.** Under the assumption of a noiseless system and a square mixing matrix in (1), the BS InfoMax is equivalent to the maximum likelihood (ML) formulation of the problem [4], which is used in this section. For the sake of simplicity of the presentation, let us consider the case where the dictionary of functions used in a source decomposition (2) is an orthonormal basis. (In this case, the corresponding coefficients $c_{mk} = <\mathbf{s}_m, \varphi_k>$, where $<\cdot, \cdot>$ denotes the inner product). From (1) and (2) the decomposition coefficients of the noiseless mixtures, according to the same signal dictionary of functions $\varphi_k(\xi)$, are:

$$\boldsymbol{\lambda}_k = \mathbf{A}\mathbf{c}_k, \tag{3}$$

where $M$-dimensional vector $\mathbf{c}_k$ forms the $k$-th column of the matrix $\mathbf{C} = \{c_{mk}\}$.

Let $\mathbf{Y}$ be the *features*, or (new) data, matrix of dimension $M \times K$, where $K$ is the number of features. Its rows are either the samples of sensor signals (mixtures), or their decomposition coefficients. In the later case, the coefficients $\boldsymbol{\lambda}_k$'s form the columns of $\mathbf{Y}$. (In the following discussion we assume this setting for $\mathbf{Y}$, if not stated other). We are interested in the maximum likelihood estimate of $\mathbf{A}$ given the data $\mathbf{Y}$.

Let the corresponding coefficients $c_{mk}$ be independent random variables with a probability density function (pdf) of an exponential type

$$p_m(c_{mk}) \propto \exp\{-\nu(c_{mk})\}, \tag{4}$$

where the scalar function $\nu(\cdot)$ is a smooth approximation of an absolute value function. Such kind of distribution is widely used for modeling sparsity [5]. In view of the independence of $c_{mk}$, and (4), the prior pdf of $\mathbf{C}$ is

$$p(\mathbf{C}) \propto \prod_{m,k} \exp\{-\nu(c_{mk})\}. \tag{5}$$

Taking into account that $\mathbf{Y} = \mathbf{AC}$, the parametric model for the pdf of $\mathbf{Y}$ with respect to parameters $\mathbf{A}$ is

$$p_{\mathbf{A}}(\mathbf{Y}) = p(\mathbf{C})/|\det A|^{K}. \tag{6}$$

Let $\mathbf{W} = \mathbf{A}^{-1}$ be the *unmixing* matrix, to be estimated. Then, substituting $\mathbf{C} = \mathbf{WY}$, combining (6) with (5) and taking the logarithm we arrive at the log-likelihood function:

$$L_{\mathbf{W}}(\mathbf{Y}) = K \log|\det \mathbf{W}| - \sum_{m=1}^{M}\sum_{k=1}^{K} \nu((\mathbf{WY})_{mk}). \tag{7}$$

Maximization of $L_{\mathbf{W}}(\mathbf{Y})$ with respect to $\mathbf{W}$ is equivalent to the BS InfoMax, and can be solved efficiently by the Natural Gradient algorithm [6]. We used this algorithm as implemented in the ICA/EEG Matlab toolbox [7].

**Clustering.** In the case of geometry based methods, separation of sparse sources can be achieved by clustering along orientations of data concentration in the $N$-dimensional space wherein each column $\mathbf{y}_k$ of the matrix $\mathbf{Y}$ represents a data point ($N$ is the number of mixtures). Let us consider a two-dimensional noiseless case, wherein two source signals, $s_1(t)$ and $s_2(t)$, are mixed by a $2{\times}2$ matrix $\mathbf{A}$, arriving at two mixtures $x_1(t)$ and $x_2(t)$. (Here, the data matrix is constructed from these mixtures $x_1(t)$ and $x_2(t)$). Typically, a scatter plot of two *sparse* mixtures $x_1(t)$ versus $x_2(t)$, looks like the rightmost plot in Figure 2. If only one source, say $s_1(t)$, was present, the sensor signals would be $x_1(t) = a_{11}s_1(t)$ and $x_2(t) = a_{21}s_1(t)$ and the data points at the scatter diagram of $x_1(t)$ versus $x_2(t)$ would belong to the straight line placed along the vector $[a_{11}a_{21}]^{T}$. The same thing happens, when two *sparse* sources are present. In this sparse case, at each particular index where a sample of the first source is large, there is a high probability, that the corresponding sample of the second source is small, and the point at the scatter diagram still lies close to the mentioned straight line. The same arguments are valid for the second source. As a result, data points are concentrated around two dominant orientations, which are directly related to the columns of $\mathbf{A}$. Source signals are rarely sparse in their original domain. In contrast, their decomposition coefficients (2) usually show high sparsity. Therefore, we construct the data matrix $\mathbf{Y}$ from the decomposition coefficients of mixtures (3), rather than from the mixtures themselves.

In order to determine orientations of scattered data, we project the data points onto the surface of a unit sphere by normalizing corresponding vectors, and then apply a standard clustering algorithm. This clustering approach works efficiently even if the number of sources is greater than the number of sensors. Our *clustering procedure* can be summarized as follows:

1. Form the feature matrix $\mathbf{Y}$, by putting samples of the sensor signals or (*subset of*) their decomposition coefficients into the corresponding rows of the matrix;

2. Normalize feature vectors (columns of $\mathbf{Y}$): $\mathbf{y}_k = \mathbf{y}_k/\|\mathbf{y}_k\|_2$, in order to project data points onto the surface of a unit sphere, where $\|\cdot\|_2$ denotes the $l_2$ norm. Before normalization, it is reasonable to remove data points with a very small norm, since these very likely to be crosstalk-corrupted by small coefficients from others' sources.

3. Move data points to a half-sphere, e.g. by forcing the sign of the first coordinate $y_k^1$ to be positive: IF $y_k^1 < 0$ THEN $\mathbf{y}_k = -\mathbf{y}_k$. Without this operation each set of linearly (i.e., along a line) clustered data points would yield two clusters on opposite sides of the sphere.

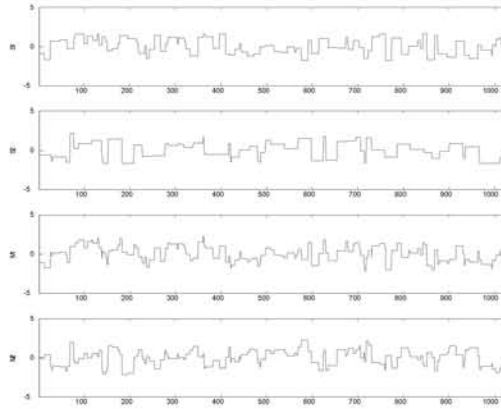

Figure 1: Random block signals (two upper) and their mixtures (two lower)

4. Estimate cluster centers by using a clustering algorithm. The coordinates of these centers will form the columns of the estimated mixing matrix $\tilde{\mathbf{A}}$. We used *Fuzzy C-means* (FCM) clustering algorithm as implemented in Matlab Fuzzy Logic Toolbox.

**Sources recovery.** The estimated unmixing matrix $\tilde{\mathbf{A}}^{-1}$ is obtained by either the BS InfoMax or the above clustering procedure, applied to either complete data set, or to some subsets of data (to be explained in the next section). Then, the sources are recovered in their original domain by $\tilde{\mathbf{s}}(t) = \tilde{\mathbf{A}}^{-1}\mathbf{x}(t)$. We should stress here that if the clustering approach is used, the estimation of sources is *not* restricted to the case of *square* mixing matrices, although the sources recovery is more complicated in the rectangular cases (this topic is out of scope of this paper).

## 3   Multinode based source separation

**Motivating example: sparsity of random blocks in the Haar basis.** To provide intuitive insight into the practical implications of our main idea, we first use 1D block functions, that are piecewise constant, with random amplitude and duration of each constant piece (Figure 1). It is known, that the Haar wavelet basis provides compact representation of such functions. Let us take a close look at the Haar wavelet coefficients at different resolution levels $j=0,1,...,J$. Wavelet basis functions at the finest resolution level $j=J$ are obtained by translation of the Haar mother wavelet: $\varphi(t) = \{1, \text{if } t \in [0,1); -1, \text{if } t \in [1,2); 0$ otherwise$\}$. Taking the scalar product of a function $s(t)$ with the wavelet $\varphi_J(t-\tau)$, we produce a finite differentiation of the function $s(t)$ at the point $t = \tau$. This means that the number of non-zero coefficients at the finest resolution for a block function will correspond roughly to the number of jumps of this function. Proceeding to the next, coarser resolution level, we have $\varphi_{J-1}(t) = \{1, \text{if } t \in [0,2); -1, \text{if } t \in [2,4); 0$ otherwise$\}$. At this level, the number of non-zero coefficients still corresponds to the number of jumps, but the total number of coefficients at this level is halved, and so is the sparsity. If we further proceed to coarser resolutions, we will encounter levels where the support of a wavelet $\varphi_j(t)$ is comparable to the typical distance between jumps in the function $s(t)$. In this case, most of the coefficients are expected to be nonzero, and, therefore, sparsity will fade away.

To demonstrate how this influences accuracy of a blind source separation, we randomly generated two block-signal sources (Figure 1, two upper plots.), and mixed them by the

crosstalk matrix $\mathbf{A}$ with columns [0.83 -0.55] and [0.62 0.78]. Resulting sensor signals, or mixtures, $x_1(t)$ and $x_2(t)$ are shown in the two lower plots of Figure 1. The scatter plot of $x_1(t)$ versus $x_2(t)$ does not exhibit any visible distinct orientations (Figure 2, left). Similarly, in the scatter plot of the wavelet coefficients at the lowest resolution distinct orientations are hardly detectable (Figure 2, middle). In contrast, the scatter plot of the wavelet coefficients at the highest resolution (Figure 2, right) depicts two distinct orientations, which correspond to the columns of the mixing matrix.

|  | Raw signals | All wavelet coefficients | High resolution WT coefficients |
|---|---|---|---|
|  |  |  |  |
| InfoMax | 1.93 | 0.183 | 0.005 |
| FCM | 1.78 | 0.058 | 0.002 |

Figure 2: Separation of block signals: scatter plots of sensor signals (left), and of their wavelet coefficients (middle and right). Lower columns present the normalized mean-squared separation error (%) corresponding to the Bell-Sejnowski InfoMax, and to the Fuzzy C-Means clustering, respectively.

Since a crosstalk matrix $\mathbf{A}$ is estimated only up to a column permutation and a scaling factor, in order to measure the separation accuracy, we normalize the original sources $s_m(t)$ and their *corresponding* estimated sources $\tilde{s}_m(t)$. The averaged (over sources) normalized squared error (NSE) is then computed as: $NSE = \frac{1}{M}\sum_{m=1}^{M}(\|\tilde{\mathbf{s}}_m - \mathbf{s}_m\|_2^2 / \|\mathbf{s}_m\|_2^2)$. Resulting separation errors for block sources are presented in the lower part of Figure 2. The largest error (1.93%) is obtained on the raw data, and the smallest ($<0.005$%) – on the wavelet coefficients at the highest resolution, which have the best sparsity. Using all wavelet coefficients yields intermediate sparsity and performance.

**Multinode representation.** Our choice of a particular wavelet basis and of the sparsest subset of coefficients was obvious in the above example: it was based on knowledge of the structure of piecewise constant signals. For sources having oscillatory components (like sounds or images with textures), other systems of basis functions, such as wavelet packets and trigonometric function libraries [9], might be more appropriate. The wavelet packet library consists of the triple-indexed family of functions: $\varphi_{j,i,q}(t) = 2^{j/2}\varphi_q(2^j t - i)$, $j, i \in \mathbf{Z}$, $q \in \mathbf{N}$,where $j, i$ are the scale and shift parameters, respectively, and $q$ is the frequency parameter. [Roughly speaking, $q$ is proportional to the number of oscillations of a mother wavelet $\varphi_q(t)$]. These functions form a binary tree whose nodes are indexed by the depth of the level $j$ and the node number $q = 0, 1, 2, 3, ..., 2^{j-1}$ at the specified level $j$. This same indexing is used for corresponding subsets of wavelet packet coefficients (as well as in scatter diagrams in the section on experimental results).

**Adaptive selection of sparse subsets.** When signals have a complex nature, it is difficult to decide in advance which nodes contain the sparsest sets of coefficients. That is why we use the following simple *adaptive approach*. First, for every node of the tree, we apply our clustering algorithm, and compute a measure of clusters' distortion. In our experiments we used a standard *global distortion*, the mean squared distance of data points to the centers of their own (closest) clusters (here again, the weights of the data points can be incorporated): $d=\sum_{k=1}^{K}\min_{m}\|\mathbf{u}_m - \mathbf{y}_k\|$,where $K$ is the number of data points, $\mathbf{u}_m$ is the $m$-th centroid coordinates, $\mathbf{y}_k$ is the $k$-th data point coordinates, and $\|.\|$ is the sum-of-squares distance.

Second, we choose a few best nodes with the minimal distortion, combine their coefficients into one data set, and apply a separation algorithm (clustering or Infomax) to these data.

## 4    Experimental results

The proposed blind separation method based on the wavelet-packet representation, was evaluated by using several types of signals. We have already discussed the relatively simple example of a random block signal. The second type of signal is a frequency modulated (FM) sinusoidal signal. The carrier frequency is modulated by either a sinusoidal function (FM signal) or by random blocks (BFM signal). The third type is a musical recording of flute sounds. Finally, we apply our algorithm to images. An example of such images is presented in the left part of Figure 3.

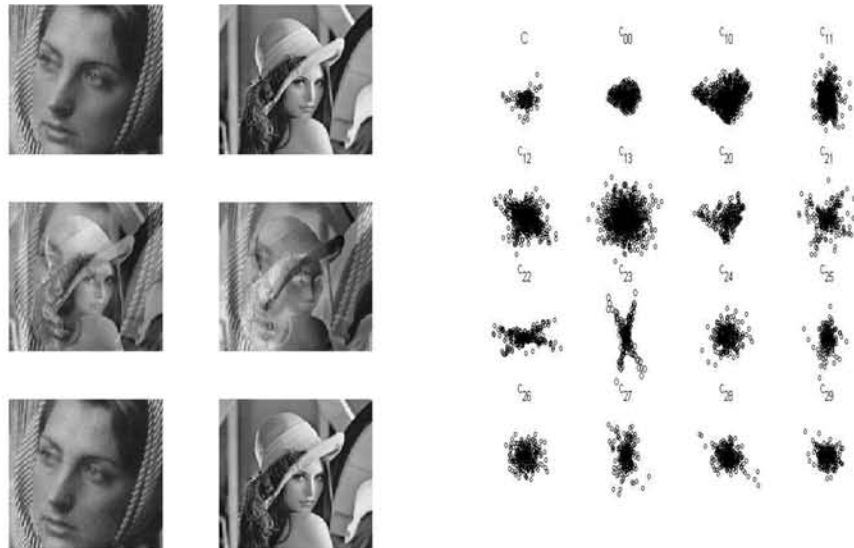

Figure 3: Left: two source images (upper pair), their mixtures (middle pair) and estimated images (lower pair). Right: scatter plots of the wavelet packet (WP) coefficients of mixtures of images; subsets are indexed on the WP tree.

In order to compare accuracy of our adaptive best nodes method with that attainable by standard methods, we form the following feature sets: (1) raw data, (2) Short Time Fourier Transform (STFT) coefficients (in the case of 1D signals), (3) Wavelet Transform coefficients (4) Wavelet packet coefficients at the best nodes found by our method, while using various wavelet families with different smoothness (haar, db-4, db-8). In the case of image separation, we used the Discrete Cosine Transform (DCT) instead of the STFT, and the sym4 and sym8 mother wavelet instead of db-4 and db-8, when using wavelet transform and wavelet packets.

The right part of Figure 3 presents an example of scatter plots of the wavelet packet coefficients obtained at various nodes of the wavelet packet tree. The upper left scatter plot, marked with 'C', corresponds to the complete set of coefficients at all nodes. The rest are the scatter plots of sets of coefficients indexed on a wavelet packet tree. Generally speaking, the more distinct the two dominant orientations appear on these plots, the more precise

is the estimation of the mixing matrix, and, therefore, the better is the quality of separation. Note, that only two nodes, $c_{22}$ and $c_{23}$, show clear orientations. These nodes will most likely be selected by the algorithm for further estimation process.

| Signals | raw data | STFT | WT db8 | WT haar | WP db8 | WP haar |
|---------|----------|------|--------|---------|--------|---------|
| Blocks | 10.16 | 2.669 | 0.174 | 0.037 | 0.073 | 0.002 |
| BFM sine | 24.51 | 0.667 | 0.665 | 2.34 | 0.2 | 0.442 |
| FM sine | 25.57 | 0.32 | 1.032 | 6.105 | 0.176 | 0.284 |
| Flutes | 1.48 | 0.287 | 0.355 | 0.852 | 0.154 | 0.648 |
| Images | raw data | DCT | WT sym8 | WT haar | WP sym8 | WP haar |
| | 4.88 | 3.651 | 1.164 | 1.114 | 0.365 | 0.687 |

Table 1: Experimental results: normalized mean-squared separation error (%) for *noise-free* signals and images, applying the FCM separation to raw data and decomposition coefficients in various domains. In the case of wavelet packets (WP) the best nodes selected by our algorithm were used.

Table 1 summarizes results of experiments in which we applied our approach of the best features selection along with the FCM separation to each noise-free feature set. In these experiments, we compared the quality of separation of deterministic signals by calculating $NSE$'s (i.e., residual crosstalk errors). In the case of random block and BFM signals, we performed 100 Monte-Carlo simulations and calculated the normalized mean-squared errors ($NMSE$) for the above feature sets. From Table 1 it is clear that using our adaptive best nodes method outperforms all other feature sets (including complete set of wavelet coefficients), for each type of signals. Similar improvement was achieved by using our method along with the BS InfoMax separation, which provided even better results for images. In the case of the random block signals, using the Haar wavelet function for the wavelet packet representation yields a better separation than using some smooth wavelet, e.g. db-8. The reason is that these block signals, that are not natural signals, have a sparser representation in the case of the Haar wavelets. In contrast, as expected, natural signals such as the Flute's signals are better represented by smooth wavelets, that in turn provide a better separation. This is another advantage of using sets of features at multiple nodes along with various families of 'mother' functions: one can choose best nodes from several decomposition trees simultaneously.

In order to verify the performance of our method in presence of noise, we added various types of noise (white gaussian and salt&pepper) to three mixtures of three images at various signal-to-noise energy ratios (SNR). Table 2 summarizes these experiments in which we applied our approach along with the BS InfoMax separation. It turns out that the ideas used in wavelet based signal denoising (see for example [10] and references therein), are applied to signal separation from *noisy mixtures*. In particular, in case of white gaussian noise, the noise energy is uniformly distributed over all wavelet coefficients at various scales. Therefore, at sufficiently high SNR's, the large coefficients of the signals are only slightly distorted by the noise coefficients, and the estimation of the unmixing matrix is almost not affected by the presence of noise. (In contrast, the BS InfoMax applied to three noisy mixtures themselves, failed completely, arriving at $NSE$ of 19% even in the case of SNR=12dB). We should stress here that, although our adaptive best nodes method performs reasonably well in the presence of noise, it is not supposed to further denoise the reconstructed images (this can be achieved by some denoising method, after source signals are separated). More experimental results, as well as parameters of simulations, can be found in [11].

| SNR [dB] | $\infty$ | 12 | 11 | 10 | 8 |
|---|---|---|---|---|---|
| Mixtures w. white gaussian noise | 0.042 | 0.192 | 0.506 | 1.628 | 17.38 |
| Mixtures w. salt&pepper noise | 0.042 | 0.047 | 0.086 | 0.240 | 2.135 |

Table 2: Performance of the algorithm in presence of various sources of noise in mixtures of images: normalized mean-squared separation error (%), applying our adaptive approach along with the BS InfoMax separation.

## 5  Conclusions

Experiments with both one- and two-dimensional simulated and natural signals demonstrate that multinode sparse representations improve the efficiency of blind source separation. The proposed method improves the separation quality by utilizing the structure of signals, wherein several subsets of the wavelet packet coefficients have significantly better sparsity and separability than others. In this case, scatter plots of these coefficients show distinct orientations each of which specifies a column of the mixing matrix. We choose the 'good subsets' according to the global distortion adopted as a measure of cluster quality. Finally, we combine together coefficients from the best chosen subsets and restore the mixing matrix using only this new subset of coefficients by the Infomax algorithm or clustering. This yields significantly better results than those obtained by applying standard Infomax and clustering approaches directly to the raw data. The advantage of our method is in particular noticeable in the case of noisy mixtures.

## Footnotes

[0]Supported in part by the Ollendorff Minerva Center, by the Israeli Ministry of Science, by NSF CAREER award 97-02-311 and by the National Foundation for Functional Brain Imaging

## References

[1] A. J. Bell and T. J. Sejnowski, "An information-maximization approach to blind separation and blind deconvolution," *Neural Computation*, vol. 7, no. 6, pp. 1129–1159, 1995.

[2] A. Hyvärinen, "Survey on independent component analysis," *Neural Computing Surveys*, no. 2, pp. 94–128, 1999.

[3] M. Zibulevsky and B. A. Pearlmutter, "Blind separation of sources with sparse representations in a given signal dictionary," *Neural Computation*, vol. 13, no. 4, pp. 863–882, 2001.

[4] J.-F. Cardoso. "Infomax and maximum likelihood for blind separation," IEEE Signal Processing Letters 4 112-114, 1997.

[5] M. S. Lewicki and T. J. Sejnowski, "Learning overcomplete representations," *Neural Computation, 12(2): 337-365, 2000.*

[6] S. Amari, A. Cichocki, and H. H. Yang, "A new learning algorithm for blind signal separation," In *Advances in Neural Information Processing Systems 8*. MIT Press. 1996.

[7] S. Makeig, ICA/EEG toolbox. Computational Neurobiology Laboratory, the Salk Institute. http://www.cnl.salk.edu/~tewon/ica_cnl.html, 1999.

[8] A. Prieto, C. G. Puntonet, and B. Prieto, "A neural algorithm for blind separation of sources based on geometric prperties.," Signal Processing, vol. 64, no. 3, pp. 315–331, 1998.

[9] S. Mallat, *A Wavelet Tour of Signal Processing*. Academic Press, 1998.

[10] D. L. Donoho, "De-Noising by Soft Thresholding," IEEE Trans. Inf. Theory, vol. 41, 3, 1995, pp. 613-627.

[11] P. Kisilev, M. Zibulevsky, Y. Y. Zeevi, and B. A. Pearlmutter, *Multiresolution framework for sparse blind source separation*, CCIT Report no.317, June 2000
